# Convergence of Laplacian Eigenmaps

**Mikhail Belkin**
Department of Computer Science
Ohio State University
Columbus, OH 43210
mbelkin@cse.ohio-state.edu

**Partha Niyogi**
Department of Computer Science
The University of Chicago
Hyde Park, Chicago, IL 60637.
niyogi@cs.uchicago.edu

## Abstract

Geometrically based methods for various tasks of machine learning have attracted considerable attention over the last few years. In this paper we show convergence of eigenvectors of the point cloud Laplacian to the eigenfunctions of the Laplace-Beltrami operator on the underlying manifold, thus establishing the first convergence results for a spectral dimensionality reduction algorithm in the manifold setting.

## 1 Introduction

The last several years have seen significant activity in geometrically motivated approaches to data analysis and machine learning. The unifying premise behind these methods is the assumption that many types of high-dimensional natural data lie on or near a low-dimensional manifold. Collectively this class of learning algorithms is often referred to as *manifold learning* algorithms. Some recent manifold algorithms include Isomap [14] and Locally Linear Embedding (LLE) [13].

In this paper we provide a theoretical analysis for the Laplacian Eigenmaps introduced in [2], a framework based on eigenvectors of the graph Laplacian associated to the point-cloud data. More specifically, we prove that under certain conditions, eigenvectors of the graph Laplacian converge to eigenfunction of the Laplace-Beltrami operator on the underlying manifold. We note that in mathematics the manifold Laplacian is a classical object of differential geometry with a rich tradition of inquiry. It is one of the key objects associated to a general differentiable Riemannian manifold. Indeed, several recent manifold learning algorithms are closely related to the Laplacian. The eigenfunction of the Laplacian are also eigenfunctions of heat diffusions, which is the point of view explored by Coifman and colleagues at Yale University in a series of recent papers on data analysis (e.g., [6]). Hessian Eigenmaps approach which uses eigenfunctions of the Hessian operator for data representation was proposed by Donoho and Grimes in [7]. Laplacian is the trace of the Hessian. Finally, as observed in [2], the cost function that is minimized to obtain the embedding of LLE is an approximation to the squared Laplacian.

In the manifold learning setting, the underlying manifold is usually unknown. Therefore functional maps from the manifold need to be estimated using point cloud data. The common approximation strategy in these methods is to construct an adjacency graph associated to a point cloud. The underlying intuition is that since the graph is a proxy for the manifold, inference based on the structure of the graph corresponds to the desired inference based on the geometric structure of the manifold. Theoretical results to justify this intuition have been developed over the last few years. Building on recent results on functional convergence of approximation for the Laplace-Beltrami operator using heat kernels and results on consistency of eigenfunctions for empirical approximations of such operators, we show convergence of the Laplacian Eigenmaps algorithm. We note that in order to prove convergence of a

spectral method, one needs to demonstrate convergence of the empirical eigenvalues and eigenfunctions. To our knowledge this is the first complete convergence proof for a spectral manifold learning method.

## 1.1 Prior and Related Work

This paper relies on results obtained in [3, 1] for functional convergence of operators. It turns out, however, that considerably more careful analysis is required to ensure spectral convergence, which is necessary to guarantee convergence of the corresponding algorithms. To the best of our knowledge previous results are not sufficient to guarantee convergence for any spectral method in the manifold setting.

Lafon in [10] generalized pointwise convergence results from [1] to the important case of an arbitrary probability distribution on the manifold. We also note [4], where a similar result is shown for the case of a domain in $\mathbb{R}^n$. Those results were further generalized and presented with an empirical pointwise convergence theorem for the manifold case in [9]. We observe that the arguments in this paper are likely to allow one to use these results to show convergence of eigenfunctions for a wide class of probability distributions on the manifold. Empirical convergence of spectral clustering for a fixed kernel parameter $t$ was analyzed in [11] and is used in this paper. However the geometric case requires $t \to 0$. The results in this paper as well as in [3, 1] are for the case of a uniform probability distribution on the manifold. Recently [8] provided deeper probabilistic analysis in that case.

Finally we point out that while the analogies between the geometry of manifolds and the geometry of graphs are well-known in spectral graph theory and in certain areas of differential geometry (see, e.g., [5]) the exact nature of that parallel is usually not made precise.

## 2 Main Result

The main result of this paper is to show convergence of eigenvectors of graph Laplacian associated to a point cloud dataset to eigenfunctions of the Laplace-Beltrami operator when the data is sampled from a uniform probability distribution on an embedded manifold.

In what follows we will assume that the manifold $\mathcal{M}$ is a compact infinitely differentiable Riemannian submanifold of $\mathbb{R}^N$ without boundary. Recall now that the Laplace-Beltrami operator $\Delta$ on $\mathcal{M}$ is a differential operator $\Delta : \mathcal{C}^2 \to L^2$ defined as

$$\Delta f = -\operatorname{div}(\nabla f)$$

where $\nabla f$ is the gradient vector field and div denotes divergence.

$\Delta$ is a positive semi-definite self-adjoint operator and has a discrete spectrum on a compact manifold. We will generally denote its $i$th smallest eigenvalue by $\lambda_i$ and the corresponding eigenfunction by $e_i$. See [12] for a thorough introduction to the subject.

We define the operator $\mathbf{L}^t : L^2(\mathcal{M}) \to L^2(\mathcal{M})$ as follows ($\mu$ is the standard measure):

$$\mathbf{L}^t(f)(p) = (4\pi t)^{-\frac{k+2}{2}} \left( \int_{\mathcal{M}} e^{-\frac{\|p-q\|^2}{4t}} f(p) \, d\mu_q - \int_{\mathcal{M}} e^{-\frac{\|p-q\|^2}{4t}} f(q) \, d\mu_q \right)$$

If $x_i$ are the data points, the corresponding empirical version is given by

$$\hat{\mathbf{L}}_n^t(f)(p) = \frac{(4\pi t)^{-\frac{k+2}{2}}}{n} \left( \sum_i e^{-\frac{\|p-x_i\|^2}{4t}} f(p) - \sum_i e^{-\frac{\|p-x_i\|^2}{4t}} f(x_i) \right)$$

The operator $\hat{\mathbf{L}}_n^t$ is (the extension of) the *point cloud Laplacian* that forms the basis of the Laplacian Eigenmaps algorithm for manifold learning. It is easy to see that it acts by matrix multiplication on functions restricted to the point cloud, with the matrix being the corresponding graph Laplacian. We will assume that $x_i$ are randomly i.i.d. sampled from $\mathcal{M}$ according to the uniform distribution.

Our main theorem shows that that there is a way to choose a sequence $t_n$, such that the eigenfunctions of the empirical operators $\hat{\mathbf{L}}_n^{t_n}$ converge to the eigenfunctions of the Laplace-Beltrami operator $\Delta$ in probability.

**Theorem 2.1** *Let $\lambda_{n,i}^t$ be the ith eigenvalue of $\hat{\mathbf{L}}_n^t$ and $e_{n,i}^t$ be the corresponding eigenfunction (which, for each fixed $i$, will be shown to exist for $t$ sufficiently small). Let $\lambda_i$ and $e_i$ be the corresponding eigenvalue and eigenfunction of $\Delta$ respectively. Then there exists a sequence $t_n \to 0$, such that*

$$\lim_{n\to\infty} \lambda_{n,i}^{t_n} = \lambda_i$$

$$\lim_{n\to\infty} \|e_{n,i}^{t_n}(x) - e_i(x)\|_2 = 0$$

*where the limits are in probability.*

## 3    Overview of the proof

The proof of the main theorem consists of two main parts. One is spectral convergence of the functional approximation $\mathbf{L}^t$ to $\Delta$ as $t \to 0$ and the other is spectral convergence of the empirical approximation $\hat{\mathbf{L}}_n^t$ to $\mathbf{L}^t$ as the number of data points $n$ tends to infinity. These two types of convergence are then put together to obtain the main Theorem 2.1.

**Part 1.** The more difficult part of the proof is to show convergence of eigenvalues and eigenfunctions of the functional approximation $\mathbf{L}^t$ to those of $\Delta$ as $t \to 0$. To demonstrate convergence we will take a different functional approximation $\frac{1-\mathbf{H}^t}{t}$ of $\Delta$, where $\mathbf{H}^t$ is the heat operator. While $\frac{1-\mathbf{H}^t}{t}$ does not converge *uniformly* to $\Delta$ they share an eigenbasis and for each *fixed* $i$ the $i$th eigenvalue of $\frac{1-\mathbf{H}^t}{t}$ converges to the $i$th eigenvalue of $\Delta$. We will then consider the operator $\mathbf{R}^t = \frac{1-\mathbf{H}^t}{t} - \mathbf{L}^t$. A careful analysis of this operator, which constitutes the bulk of the proof paper, shows that $\mathbf{R}^t$ is a small *relatively bounded* perturbation of $\frac{1-\mathbf{H}^t}{t}$, in the sense that for any function $f$ we have $\frac{\|\mathbf{R}^t f\|_2}{\|\frac{1-\mathbf{H}^t}{t} f\|_2} \ll 1$ as $t \to 0$. This will imply spectral convergence and lead to the following

**Theorem 3.1** *Let $\lambda_i, \lambda_i^t, e_i, e_i^t$ be the ith smallest eigenvalues and the corresponding eigenfunctions of $\Delta$ and $\mathbf{L}^t$ respectively. Then*

$$\lim_{t\to 0} |\lambda_i - \lambda_i^t| = 0$$

$$\lim_{t\to 0} \|e_i - e_i^t\|_2 = 0$$

**Part 2.** The second part is to show that the eigenfunctions of the empirical operator $\hat{\mathbf{L}}_n^t$ converge to eigenfunctions of $\mathbf{L}^t$ as $n \to \infty$ in probability. That result follows readily from the previous work in [11] together with the analysis of the essential spectrum of $\mathbf{L}^t$. The following theorem is obtained:

**Theorem 3.2** *For a fixed sufficiently small $t$, let $\lambda_{n,i}^t$ and $\lambda_i^t$ be the ith eigenvalue of $\hat{\mathbf{L}}_n^t$ and $L^t$ respectively. Let $e_{n,i}^t$ and $e_i^t$ be the corresponding eigenfunctions. Then*

$$\lim_{n\to\infty} \lambda_{n,i}^t = \lambda_i^t$$

$$\lim_{n\to\infty} \|e_{n,i}^t(x) - e_i^t(x)\|_2 = 0$$

*assuming that $\lambda_i^t \le \frac{1}{2t}$. The convergence is almost sure.*

Observe that this implies convergence for any fixed $i$ as soon as $t$ is sufficiently small.

Symbolically these two theorems can be represented by top line of the following diagram:

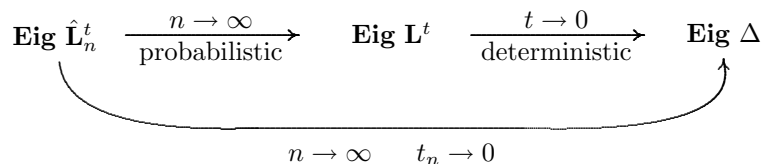

After demonstrating two types of convergence results in the top line of the diagram a simple argument shows that a sequence $t_n$ can be chosen to guarantee convergence as in the final Theorem 2.1 and provides the bottom arrow.

# 4  Spectral Convergence of Functional Approximations.

## 4.1  Main Objects and the Outline of the Proof

Let $\mathcal{M}$ be a compact smooth smoothly embedded $k$-dimensional manifold in $\mathbb{R}^N$ with the induced Riemannian structure and the corresponding induced measure $\mu$.

As above, we define the operator $\mathbf{L}^t : L^2(\mathcal{M}) \to L^2(\mathcal{M})$ as follows:

$$\mathbf{L}^t(f)(x) = (4\pi t)^{-\frac{k+2}{2}} \left( \int_{\mathcal{M}} e^{-\frac{\|x-y\|^2}{4t}} f(x) \, d\mu_y - \int_{\mathcal{M}} e^{-\frac{\|x-y\|^2}{4t}} f(y) \, d\mu_y \right)$$

As shown in previous work, this operator serves as a functional approximation to the Laplace-Beltrami operator on $\mathcal{M}$. The purpose of this paper is to extend the previous results to the eigenvalues and eigenfunctions, which turn out to need some careful estimates.

We start by reviewing certain properties of the Laplace-Beltrami operator and its connection to the heat equation. Recall that the heat equation on the manifold $\mathcal{M}$ is given by

$$\Delta h(x,t) = \frac{\partial h(x,t)}{\partial t}$$

where $h(x,t)$ is the heat at time $t$ at point $x$. Let $f(x) = h(x,0)$ be the initial heat distribution. We observe that from the definition of the derivative

$$\Delta f = \lim_{t \to 0} \frac{1}{t}(h(x,t) - f(x))$$

It is well-known (e.g., [12]) that the solution to the heat equation at time $t$ can be written as

$$\mathbf{H}^t f(x) := h(x,t) = \int_{\mathcal{M}} H^t(x,y) f(y) d\mu_y$$

Here $\mathbf{H}^t$ is the *heat operator* and $H^t(x,y)$ is the *heat kernel* of $\mathcal{M}$. It is also well-known that the heat operator $\mathbf{H}^t$ can be written as $\mathbf{H}^t = e^{-t\Delta}$. We immediately see that $\Delta = \lim_{t\to 0} \frac{1-\mathbf{H}^t}{t}$ and that eigenfunctions of $\mathbf{H}^t$ and hence eigenfunction of $\frac{1-\mathbf{H}^t}{t}$ coincide with eigenfunctions of the Laplace operator. The $i$th eigenvalue of $\frac{1-\mathbf{H}^t}{t}$ is equal to $\frac{1-e^{-t\lambda_i}}{t}$, where $\lambda_i$ as usual is the $i$th eigenvalue of $\Delta$.

It is easy to observe that once the heat kernel $H^t(x,y)$ is known, finding the Laplace operator poses no difficulty:

$$\Delta f = \lim_{t \to 0} \frac{1}{t} \left( f(x) - \int_{\mathcal{M}} H^t(x,y) f(y) \, d\mu_y \right) = \lim_{t \to 0} \left( \frac{1 - \mathbf{H}^t}{t} \right) f \qquad (1)$$

Reconstructing the Laplacian from a point cloud is possible because of the fundamental fact that the manifold heat kernel $H^t(x,y)$ can be approximated by the ambient space Gaussian and hence $\mathbf{L}^t$ is an approximation to $\frac{1-\mathbf{H}^t}{t}$ and can be shown to converge for a fixed $f$ to $\Delta$. This pointwise operator convergence is discussed in [10, 3, 1].

To obtain convergence of eigenfunctions, however, one typically needs the stronger *uniform convergence*. If $\mathbf{A}_n$ is a sequence of operators, we say that $\mathbf{A}_n \to \mathbf{A}$ uniformly in $L^2$ if $\sup_{\|f\|_2=1} \|\mathbf{A}_n f - \mathbf{A} f\|_2 \to 0$. This is sufficient for convergence of eigenfunctions and other spectral properties.

It turns out that this type of convergence *does not* hold for functional approximation $\mathbf{L}^t$ as $t \to 0$, which presents a serious technical obstruction to proving convergence of spectral properties. To observe that $\mathbf{L}^t$ does not converge uniformly to $\Delta$, observe that while $\frac{1-\mathbf{H}^t}{t}$

converges to $\Delta$ for each fixed function $f$, even this convergence is not uniform. Indeed, for a small $t$, we can always choose a sufficiently large $\lambda_i \gg 1/t$ and the corresponding eigenfunction $e_i$ of $\Delta$, s.t.

$$\left\| \left( \frac{1 - \mathbf{H}^t}{t} - \Delta \right) e_i \right\|_2 = \left| \frac{1}{t}(1 - e^{-t\lambda_i}) - \lambda_i \right| \approx \left| \frac{1}{t} - \lambda_i \right| \gg 1$$

Since $\mathbf{L}^t$ is an approximation to $\frac{1 - \mathbf{H}^t}{t}$, uniform convergence cannot be expected and the standard perturbation theory techniques do not apply. To overcome this obstacle we need the two following key ingredients:

**Observation 1.** Eigenfunctions of $\frac{1 - \mathbf{H}^t}{t}$ coincide with eigenfunctions of $\Delta$.

**Observation 2.** $\mathbf{L}^t$ is a small *relatively bounded* perturbation of $\frac{1 - \mathbf{H}^t}{t}$.

While the first of these observations is immediate, the second is the technical core of this work. The *relative boundedness* of the perturbation will imply convergence of eigenfunctions of $\mathbf{L}^t$ to those of $\frac{1 - \mathbf{H}^t}{t}$ and hence, by the Observation 1, to eigenfunctions of $\Delta$.

We now define the perturbation operator

$$\mathbf{R}^t = \frac{1 - \mathbf{H}^t}{t} - \mathbf{L}^t$$

The relative boundedness of the self-adjoint perturbation operator $\mathbf{R}^t$ is formalized as follows:

**Theorem 4.1** *For any $0 < \epsilon < \frac{2}{k+2}$ there exists a constant $C$, such that for all $t$ sufficiently small*

$$\frac{|\langle \mathbf{R}^t f, f \rangle|}{\langle \frac{1 - \mathbf{H}^t}{t} f, f \rangle} \leq C \max\left( t^{\frac{2}{k+2} - \epsilon}, t^{\frac{k+2}{2}\epsilon} \right)$$

*In particular*

$$\lim_{t \to 0} \sup_{\|f\|_2 = 1} \frac{\langle \mathbf{R}^t f, f \rangle}{\langle \frac{1 - \mathbf{H}^t}{t} f, f \rangle} = 0$$

*and hence $\mathbf{R}^t$ is dominated by $\frac{1 - \mathbf{H}^t}{t}$ on $L^2$ as $t$ tends to $0$.*

This result implies that for small values of $t$, bottom eigenvalues and eigenfunction of $\mathbf{L}^t$ are close to those of $\frac{1 - \mathbf{H}^t}{t}$, which in turn implies convergence. To establish this result, we will need two key estimates on the size of the perturbation $\mathbf{R}^t$ in two different norms.

**Proposition 4.2** *Let $f \in L^2$. There exists $C \in \mathbb{R}$, such that for all sufficiently small values of $t$*

$$\|\mathbf{R}^t f\|_2 \leq C\|f\|_2$$

**Proposition 4.3** *Let $f \in H^{\frac{k}{2}+1}$, where $H^{\frac{k}{2}+1}$ is a Sobolev space. Then there is $C \in \mathbb{R}$, such that for all sufficiently small values of $t$*

$$\|\mathbf{R}^t f\|_2 \leq C\sqrt{t}\|f\|_{H^{\frac{k}{2}+1}}$$

In what follows we give the proof of the Theorem 4.1 assuming the two Propositions above. The proof of the Propositions requires technical estimates of the heat kernel and can be found the longer version of the paper enclosed.

## 4.2   Proof of Theorem 4.1.

**Lemma 4.4** *Let $e$ be an eigenvector of $\Delta$ with the eigenvalue $\lambda$. Then for some universal constant $C$*

$$\|e\|_{H^{\frac{k}{2}+1}} \leq C\lambda^{\frac{k+2}{4}} \tag{2}$$

The details can be found in the long version. Now we can proceed with the

PROOF: [Theorem 4.1]

Let $e_i(x)$ be the $i$th eigenfunction of $\Delta$ and let $\lambda_i$ be the corresponding eigenvalue. Recall that $e_i$ form an orthonormal basis of $L^2(\mathcal{M})$. Thus any function $f \in L^2(\mathcal{M})$ can be written uniquely as $f(x) = \sum_{i=0}^{\infty} a_i e_i(x)$ where $\sum a_i^2 < \infty$. For technical resons we will assume that all our functions are perpendicular to the constant and the lowest eigenvalue is nonzero.

Recall also that

$$\mathbf{H}^t f = \exp(-t\Delta)f, \qquad \mathbf{H}^t e_i = \exp(-t\lambda_i)e_i, \qquad \frac{1 - \mathbf{H}^t}{t}e_i = \frac{1 - e^{-\lambda_i t}}{t}e_i \qquad (3)$$

Now let us fix $t$ and consider the function $\phi(x) = \frac{1 - e^{-xt}}{t}$ for positive $x$. It is easy to check that $\phi$ is a concave and increasing function of $x$.

Put $x_0 = 1/\sqrt{t}$. We have:

$$\phi(0) = 0 \qquad \phi(x_0) = \frac{1 - e^{-\sqrt{t}}}{t} \qquad \frac{\phi(x_0)}{x_0} = \frac{1 - e^{-\sqrt{t}}}{\sqrt{t}}$$

Splitting the positive real line in two intervals $[0, x_0]$, $[x_0, \infty)$ and using concavity and monotonicity we observe that

$$\phi(x) \geq \min\left(\frac{1 - e^{-\sqrt{t}}}{\sqrt{t}}x, \frac{1 - e^{-\sqrt{t}}}{t}\right)$$

Note that $\lim_{t \to 0} \frac{1 - e^{-\sqrt{t}}}{\sqrt{t}} = 1$.

Therefore for $t$ sufficiently small

$$\phi(x) \geq \min\left(\frac{1}{2}x, \frac{1}{2\sqrt{t}}\right)$$

Thus

$$\left\langle \frac{1 - \mathbf{H}^t}{t}e_i, e_i \right\rangle = \frac{1 - e^{-\lambda_i t}}{t} \geq \frac{1}{2}\min\left(\lambda_i, \frac{1}{\sqrt{t}}\right) \qquad (4)$$

Now take $f \in L^2$, $f(x) = \sum_1^{\infty} a_i e_i(x)$. Without a loss of generality we can assume that $\|f\|_2 = 1$. Taking $\alpha > 0$, we split $f$ as a sum of $f_1$ and $f_2$ as following:

$$f_1 = \sum_{\lambda_i \leq \alpha} a_i e_i, \qquad f_2 = \sum_{\lambda_i > \alpha} a_i e_i$$

It is clear that $f = f_1 + f_2$ and, since $f_1$ and $f_2$ are orthogonal, $\|f\|_2^2 = \|f_1\|_2^2 + \|f_2\|_2^2$. We will now deal separately with $f_1$ and with $f_2$.

From the inequality (4) above, we observe that

$$\left\langle \frac{1 - \mathbf{H}^t}{t}f, f \right\rangle \geq \frac{1}{2}\lambda_1$$

On the other hand, from the inequality (2), we see that if $e_i$ is a basis element present in the basis expansion of $f_1$,

$$\|e_i\|_H^{\frac{k}{2}+1} \leq C\alpha^{\frac{k+2}{4}}$$

Since $\Delta$ acts by rescaling basis elements, we have $\|f_1\|_{H^{\frac{k}{2}+1}} \leq C\alpha^{\frac{k+2}{4}}$.

Therefore by Proposition 4.3 for $t$ sufficiently small and some constant $C'$

$$\|\mathbf{R}^t f_1\|_2 \leq C'\sqrt{t}\alpha^{\frac{k+2}{4}} \qquad (5)$$

Hence we see that

$$\frac{\|\mathbf{R}^t f_1\|_2}{\langle \frac{1-\mathbf{H}^t}{t} f, f \rangle} \leq \frac{2C'}{\lambda_1} \sqrt{t}\, \alpha^{\frac{k+2}{4}} \tag{6}$$

Consider now the second summand $f_2$. Recalling that $f_2$ only has basis components with eigenvalues greater than $\alpha$ and using the inequality (4) we see that

$$\left\langle \frac{1-\mathbf{H}^t}{t} f, f \right\rangle \geq \left\langle \frac{1-\mathbf{H}^t}{t} f_2, f_2 \right\rangle \geq \frac{1}{2} \min\left(\alpha, \frac{1}{\sqrt{t}}\right) \|f_2\|_2^2 \tag{7}$$

On the other hand, by Proposition 4.2

$$\|\mathbf{R}^t f_2\|_2 \leq C_1 \|f_2\|_2^2 \tag{8}$$

Thus

$$\frac{|\langle \mathbf{R}^t f_2, f_2 \rangle|}{\langle \frac{1-\mathbf{H}^t}{t} f, f \rangle} \leq \frac{\|\mathbf{R}^t f_2\|_2}{\langle \frac{1-\mathbf{H}^t}{t} f_2, f_2 \rangle} \leq C_1' \max\left(\frac{1}{\alpha}, \sqrt{t}\right) \tag{9}$$

Finally, collecting inequalities 6 and 9 we see:

$$\frac{|\langle \mathbf{R}^t f, f \rangle|}{\langle \frac{1-\mathbf{H}^t}{t} f, f \rangle} \leq \frac{\|\mathbf{R}^t f_1\| + \|\mathbf{R}^t f_2\|}{\langle \frac{1-\mathbf{H}^t}{t} f, f \rangle} \leq C\left(\max\left(\frac{1}{\alpha}, \sqrt{t}\right) + \sqrt{t}\, \alpha^{\frac{k+2}{4}}\right) \tag{10}$$

where $C$ is a constant independent of $t$ and $\alpha$.

Choosing $\alpha = t^{-\frac{2}{k+2}+\epsilon}$ where $0 < \epsilon < \frac{2}{k+2}$ yields the desired result. $\square$

## 5  Spectral Convergence of Empirical Approximation

**Proposition 5.1** *For $t$ sufficiently small*

$$\text{SpecEss}\,(\mathbf{L}^t) \subset \left(\frac{1}{2} t^{-1}, \infty\right)$$

*where* SpecEss *denotes the essential spectrum of the operator.*

PROOF: As noted before $\mathbf{L}^t f$ is a difference of a multiplication operator and a compact operator

$$\mathbf{L}^t f(p) = g(p) f(p) - Kf \tag{11}$$

where

$$g(p) = (4\pi t)^{-\frac{k+2}{2}} \int_{\mathcal{M}} e^{-\frac{\|p-q\|^2}{4t}}\, d\mu_q$$

and $Kf$ is a convolution with a Gaussian. As noted in [11], it is a fact in basic perturbation theory SpecEss $(\mathbf{L}^t) = \text{rg}\, g$ where $\text{rg}\, g$ is the range of the function $g : \mathcal{M} \to \mathbb{R}$. To estimate $\text{rg}\, g$ observe first that

$$\lim_{t\to\infty} (4\pi t)^{-\frac{k}{2}} \int_{\mathcal{M}} e^{-\frac{\|p-q\|^2}{4t}}\, d\mu_q = 1$$

We thus see that for $t$ sufficiently small

$$(4\pi t)^{-\frac{k}{2}} \int_{\mathcal{M}} e^{-\frac{\|p-y\|^2}{4t}}\, d\mu_y > \frac{1}{2}$$

and hence $g(t) > \frac{1}{2} t^{-1}$. $\square$

**Lemma 5.2** *Let $e^t$ be an eigenfunction of $\mathbf{L}^t$, $\mathbf{L}^t e^t = \lambda^t e^t$, $\lambda^t < \frac{1}{2} t^{-1}$. Then $e^t \in \mathcal{C}^\infty$.*

We see that Theorem 3.2 follows easily:

PROOF: [Theorem 3.2] By the Proposition 5.1 we see that the part of the spectrum of $\mathbf{L}^t$ between 0 and $\frac{1}{2} t^{-1}$ is discrete. It is a standard fact of functional analysis that such points are eigenvalues and there are corresponding eigenspaces of finite dimension. Consider now $\lambda_i^t \in [0, \frac{1}{2} t^{-1}]$ and the corresponding eigenfunction $e_i^t$. The Theorem 4 then follows from Theorem 23 and Proposition 25 in [11], which show convergence of spectral properties for the empirical operators. $\square$

## 6    Main Theorem

We are finally in position to prove the main Theorem 4.1: PROOF: [Theorem 4.1] From Theorems 3.2 and 3.1 we obtain the following convergence results:

$$\mathbf{Eig}\ \hat{\mathbf{L}}_n^t \quad \xrightarrow{\ n \to \infty\ } \quad \mathbf{Eig}\ L^t \quad \xrightarrow{\ t \to 0\ } \quad \mathbf{Eig}\ \Delta$$

where the first convergence is almost surely for $\lambda_i \leq \frac{1}{2}t^{-1}$. Given any $i \in \mathbb{N}$ and any $\epsilon > 0$, we can choose $t' < 2\lambda_i^{-1}$, s.t. for all $t < t'$ we have $\|e_i - e_i^t\|_2 < \frac{\epsilon}{2}$. On the other hand, by using the first arrow, we see that

$$\lim_{n \to \infty} \mathbb{P}\left\{\|e_{n,i}^t - e_i^t\|_2 \geq \frac{\epsilon}{2}\right\} = 0$$

Thus for any $p > 0$ and for each $t$ there exists an $N$, s.t. $\mathbb{P}\{\|e_{n,i}^t - e_i\|_2 > \epsilon\} < p$ Inverting this relationship, we see that for any $N$ and for any probability $p(N)$ there exists a $t_N$, s.t.

$$\forall_{n > N} \quad \mathbb{P}\{\|e_{n,i}^{t_N} - e_i\|_2 > \epsilon\} < p(N)$$

Making $p(N)$ tend to zero, we obtain convergence in probability.    □

## References

[1] M. Belkin, *Problems of Learning on Manifolds*, Univ. of Chicago, Ph.D. Diss., 2003.

[2] M. Belkin, P. Niyogi, *Laplacian Eigenmaps and Spectral Techniques for Embedding and Clustering*, NIPS 2001.

[3] M. Belkin, P. Niyogi, *Towards a Theoretical Foundation for Laplacian-Based Manifold Methods*, COLT 2005.

[4] O. Bousquet, O. Chapelle, M. Hein, *Measure Based Regularization*, NIPS 2003.

[5] F. R. K. Chung. (1997). *Spectral Graph Theory*. Regional Conference Series in Mathematics, number 92.

[6] R.R.Coifman, S. Lafon, A. Lee, M. Maggioni, B. Nadler, F. Warner and S. Zucker, *Geometric diffusions as a tool for harmonic analysis and structure definition of data*, submitted to the Proceedings of the National Academy of Sciences (2004).

[7] D. L. Donoho, C. E. Grimes, *Hessian Eigenmaps: new locally linear embedding techniques for high-dimensional data*, PNAS, vol. 100 pp. 5591-5596.

[8] E. Gine, V. Kolchinski, *Empirical Graph Laplacian Approximation of Laplace-Beltrami Operators: Large Sample Results*, preprint.

[9] M. Hein, J.-Y. Audibert, U. von Luxburg, *From Graphs to Manifolds – Weak and Strong Pointwise Consistency of Graph Laplacians*, COLT 2005.

[10] S. Lafon, *Diffusion Maps and Geodesic Harmonics*, Ph.D.Thesis, Yale University, 2004.

[11] U. von Luxburg, M. Belkin, O. Bousquet, *Consistency of Spectral Clustering*, Max Planck Institute for Biological Cybernetics Technical Report TR 134, 2004.

[12] S. Rosenberg, *The Laplacian on a Riemannian Manifold*, Cambridge Univ. Press, 1997.

[13] Sam T. Roweis, Lawrence K. Saul. (2000). *Nonlinear Dimensionality Reduction by Locally Linear Embedding*, Science, vol 290.

[14] J.B.Tenenbaum, V. de Silva, J. C. Langford. (2000). *A Global Geometric Framework for Nonlinear Dimensionality Reduction*, Science, Vol 290.
